# A Denoising View of Matrix Completion

**Weiran Wang**  **Miguel Á. Carreira-Perpiñán**
EECS, University of California, Merced
http://eecs.ucmerced.edu

**Zhengdong Lu**
Microsoft Research Asia, Beijing
zhengdol@microsoft.com

## Abstract

In matrix completion, we are given a matrix where the values of only some of the entries are present, and we want to reconstruct the missing ones. Much work has focused on the assumption that the data matrix has low rank. We propose a more general assumption based on denoising, so that we expect that the value of a missing entry can be predicted from the values of neighboring points. We propose a nonparametric version of denoising based on local, iterated averaging with mean-shift, possibly constrained to preserve local low-rank manifold structure. The few user parameters required (the denoising scale, number of neighbors and local dimensionality) and the number of iterations can be estimated by cross-validating the reconstruction error. Using our algorithms as a postprocessing step on an initial reconstruction (provided by e.g. a low-rank method), we show consistent improvements with synthetic, image and motion-capture data.

Completing a matrix from a few given entries is a fundamental problem with many applications in machine learning, computer vision, network engineering, and data mining. Much interest in matrix completion has been caused by recent theoretical breakthroughs in compressed sensing [1, 2] as well as by the now celebrated Netflix challenge on practical prediction problems [3, 4]. Since completion of arbitrary matrices is not a well-posed problem, it is often assumed that the underlying matrix comes from a restricted class. Matrix completion models almost always assume a low-rank structure of the matrix, which is partially justified through factor models [4] and fast convex relaxation [2], and often works quite well when the observations are sparse and/or noisy. The low-rank structure of the matrix essentially asserts that all the column vectors (or the row vectors) live on a low-dimensional subspace. This assumption is arguably too restrictive for problems with richer structure, e.g. when each column of the matrix represents a snapshot of a seriously corrupted motion capture sequence (see section 3), for which a more flexible model, namely a curved manifold, is more appropriate.

In this paper, we present a novel view of matrix completion based on manifold denoising, which conceptually generalizes the low-rank assumption to curved manifolds. Traditional manifold denoising is performed on fully observed data [5, 6], aiming to send the data corrupted by noise back to the correct surface (defined in some way). However, with a large proportion of missing entries, we may not have a good estimate of the manifold. Instead, we start with a poor estimate and improve it iteratively. Therefore the "noise" may be due not just to intrinsic noise, but mostly to inaccurately estimated missing entries. We show that our algorithm can be motivated from an objective purely based on denoising, and prove its convergence under some conditions. We then consider a more general case with a nonlinear low-dimensional manifold and use a stopping criterion that works successfully in practice. Our model reduces to a low-rank model when we require the manifold to be flat, showing a relation with a recent thread of matrix completion models based on alternating projection [7]. In our experiments, we show that our denoising-based matrix completion model can make better use of the latent manifold structure on both artificial and real-world data sets, and yields superior recovery of the missing entries.

The paper is organized as follows: section 1 reviews nonparametric denoising methods based on mean-shift updates, section 2 extends this to matrix completion by using denoising with constraints, section 3 gives experimental results, and section 4 discusses related work.

# 1 Denoising with (manifold) blurring mean-shift algorithms (GBMS/MBMS)

In Gaussian blurring mean-shift (GBMS), denoising is performed in a nonparametric way by local averaging: each data point moves to the average of its neighbors (to a certain scale), and the process is repeated. We follow the derivation in [8]. Consider a dataset $\{\mathbf{x}_n\}_{n=1}^N \subset \mathbb{R}^D$ and define a Gaussian kernel density estimate

$$p(\mathbf{x}) = \frac{1}{N} \sum_{n=1}^N G_\sigma(\mathbf{x}, \mathbf{x}_n) \tag{1}$$

with bandwidth $\sigma > 0$ and kernel $G_\sigma(\mathbf{x}, \mathbf{x}_n) \propto \exp\left(-\frac{1}{2}(\|\mathbf{x} - \mathbf{x}_n\|/\sigma)^2\right)$ (other kernels may be used, such as the Epanechnikov kernel, which results in sparse affinities). The (non-blurring) *mean-shift algorithm* rearranges the stationary point equation $\nabla p(\mathbf{x}) = \mathbf{0}$ into the iterative scheme $\mathbf{x}^{(\tau+1)} = \mathbf{f}(\mathbf{x}^{(\tau)})$ with

$$\mathbf{x}^{(\tau+1)} = \mathbf{f}(\mathbf{x}^{(\tau)}) = \sum_{n=1}^N p(n|\mathbf{x}^{(\tau)})\mathbf{x}_n \quad p(n|\mathbf{x}^{(\tau)}) = \frac{\exp\left(-\frac{1}{2}\left\|(\mathbf{x}^{(\tau)} - \mathbf{x}_n)/\sigma\right\|^2\right)}{\sum_{n'=1}^N \exp\left(-\frac{1}{2}\left\|(\mathbf{x}^{(\tau)} - \mathbf{x}_{n'})/\sigma\right\|^2\right)}. \tag{2}$$

This converges to a mode of $p$ from almost every initial $\mathbf{x} \in \mathbb{R}^D$, and can be seen as taking self-adapting step sizes along the gradient (since the *mean shift* $\mathbf{f}(\mathbf{x}) - \mathbf{x}$ is parallel to $\nabla p(\mathbf{x})$). This iterative scheme was originally proposed by [9] and it or variations of it have found widespread application in clustering [8, 10–12] and denoising of 3D point sets (surface fairing; [13, 14]) and manifolds in general [5, 6].

The *blurring mean-shift algorithm* applies one step of the previous scheme, initialized from every point, in parallel for all points. That is, given the dataset $\mathbf{X} = \{\mathbf{x}_1, \ldots, \mathbf{x}_N\}$, for each $\mathbf{x}_n \in \mathbf{X}$ we obtain a new point $\tilde{\mathbf{x}}_n = \mathbf{f}(\mathbf{x}_n)$ by applying one step of the mean-shift algorithm, and then we replace $\mathbf{X}$ with the new dataset $\tilde{\mathbf{X}}$, which is a blurred (shrunk) version of $\mathbf{X}$. By iterating this process we obtain a sequence of datasets $\mathbf{X}^{(0)}, \mathbf{X}^{(1)}, \ldots$ (and a corresponding sequence of kernel density estimates $p^{(0)}(\mathbf{x}), p^{(1)}(\mathbf{x}), \ldots$) where $\mathbf{X}^{(0)}$ is the original dataset and $\mathbf{X}^{(\tau)}$ is obtained by blurring $\mathbf{X}^{(\tau-1)}$ with one mean-shift step. We can see this process as maximizing the following objective function [10] by taking parallel steps of the form (2) for each point:

$$E(\mathbf{X}) = \sum_{n=1}^N p(\mathbf{x}_n) = \frac{1}{N} \sum_{n,m=1}^N G_\sigma(\mathbf{x}_n, \mathbf{x}_m) \propto \sum_{n,m=1}^N e^{-\frac{1}{2}\left\|\frac{\mathbf{x}_n - \mathbf{x}_m}{\sigma}\right\|^2}. \tag{3}$$

This process eventually converges to a dataset $\mathbf{X}^{(\infty)}$ where all points are coincident: a completely denoised dataset where all structure has been erased. As shown by [8], this process can be stopped early to return clusters (= locally denoised subsets of points); the number of clusters obtained is controlled by the bandwidth $\sigma$. However, here we are interested in the denoising behavior of GBMS.

The GBMS step can be formulated in a matrix form reminiscent of spectral clustering [8] as $\tilde{\mathbf{X}} = \mathbf{X}\mathbf{P}$ where $\mathbf{X} = (\mathbf{x}_1, \ldots, \mathbf{x}_N)$ is a $D \times N$ matrix of data points; $\mathbf{W}$ is the $N \times N$ matrix of Gaussian affinities $w_{nm} = G_\sigma(\mathbf{x}_n, \mathbf{x}_m)$; $\mathbf{D} = \text{diag}\left(\sum_{n=1}^N w_{nm}\right)$ is the degree matrix; and $\mathbf{P} = \mathbf{W}\mathbf{D}^{-1}$ is an $N \times N$ stochastic matrix: $p_{nm} = p(n|\mathbf{x}_m) \in (0, 1)$ and $\sum_{n=1}^N p_{nm} = 1$. $\mathbf{P}$ (or rather its transpose) is the stochastic matrix of the random walk in a graph [15], which in GBMS represents the posterior probabilities of each point under the kernel density estimate (1). $\mathbf{P}$ is similar to the matrix $\mathbf{N} = \mathbf{D}^{-\frac{1}{2}}\mathbf{W}\mathbf{D}^{-\frac{1}{2}}$ derived from the normalized graph Laplacian commonly used in spectral clustering, e.g. in the normalized cut [16]. Since, by the Perron-Frobenius theorem [17, ch. 8], all left eigenvalues of $\mathbf{P}(\mathbf{X})$ have magnitude less than 1 except for one that equals 1 and is associated with an eigenvector of constant entries, iterating $\tilde{\mathbf{X}} = \mathbf{X}\mathbf{P}(\mathbf{X})$ converges to the stationary distribution of each $\mathbf{P}(\mathbf{X})$, where all points coincide.

From this point of view, the product $\tilde{\mathbf{X}} = \mathbf{X}\mathbf{P}(\mathbf{X})$ can be seen as filtering the dataset $\mathbf{X}$ with a data-dependent low-pass filter $\mathbf{P}(\mathbf{X})$, which makes clear the denoising behavior. This also suggests using other filters [12] $\tilde{\mathbf{X}} = \mathbf{X}\phi(\mathbf{P}(\mathbf{X}))$ as long as $\phi(1) = 1$ and $|\phi(r)| < 1$ for $r \in [0, 1)$, such as explicit schemes $\phi(\mathbf{P}) = (1 - \eta)\mathbf{I} + \eta\mathbf{P}$ for $\eta \in (0, 2]$, power schemes $\phi(\mathbf{P}) = \mathbf{P}^n$ for $n = 1, 2, 3 \ldots$ or implicit schemes $\phi(\mathbf{P}) = ((1 + \eta)\mathbf{I} - \eta\mathbf{P})^{-1}$ for $\eta > 0$.

One important problem with GBMS is that it denoises equally in all directions. When the data lies on a low-dimensional manifold, denoising orthogonally to it removes out-of-manifold noise, but

denoising tangentially to it perturbs intrinsic degrees of freedom of the data and causes shrinkage of the entire manifold (most strongly near its boundary). To prevent this, the *manifold blurring mean-shift algorithm (MBMS)* [5] first computes a predictor averaging step with GBMS, and then for each point $\mathbf{x}_n$ a corrector projective step removes the step direction that lies in the local tangent space of $\mathbf{x}_n$ (obtained from local PCA run on its $k$ nearest neighbors). In practice, both GBMS and MBMS must be stopped early to prevent excessive denoising and manifold distortions.

## 2  Blurring mean-shift denoising algorithms for matrix completion

We consider the natural extension of GBMS to the matrix completion case by adding the constraints given by the present values. We use the subindex notation $\mathbf{X}_\mathcal{M}$ and $\mathbf{X}_\mathcal{P}$ to indicate selection of the missing or present values of the matrix $\mathbf{X}_{D \times N}$, where $\mathcal{P} \subset \mathcal{U}$, $\mathcal{M} = \mathcal{U} \setminus \mathcal{P}$ and $\mathcal{U} = \{(d, n) \colon d = 1, \dots, D, \ n = 1, \dots, N\}$. The indices $\mathcal{P}$ and values $\overline{\mathbf{X}}_\mathcal{P}$ of the present matrix entries are the data of the problem. Then we have the following constrained optimization problem:

$$\max_{\mathbf{X}} E(\mathbf{X}) = \sum_{n,m=1}^{N} G_\sigma(\mathbf{x}_n, \mathbf{x}_m) \quad \text{s.t.} \quad \mathbf{X}_\mathcal{P} = \overline{\mathbf{X}}_\mathcal{P}. \tag{4}$$

This is similar to low-rank formulations for matrix completion that have the same constraints but use as objective function the reconstruction error with a low-rank assumption, e.g. $\|\mathbf{X} - \mathbf{ABX}\|^2$ with $A_{D \times L}$, $B_{L \times D}$ and $L < D$.

We initialize $\mathbf{X}_\mathcal{M}$ to the output of some other method for matrix completion, such as singular value projection (SVP; [7]). For simple constraints such as ours, gradient projection algorithms are attractive. The gradient of $E$ wrt $\mathbf{X}$ is a matrix of $D \times N$ whose $n$th column is:

$$\nabla_{\mathbf{x}_n} E(\mathbf{X}) = \frac{2}{\sigma^2} \sum_{m=1}^{N} e^{-\frac{1}{2}\left\|\frac{\mathbf{x}_n - \mathbf{x}_m}{\sigma}\right\|^2} (\mathbf{x}_m - \mathbf{x}_n) \propto \frac{2}{\sigma^2} p(\mathbf{x}_n) \left( -\mathbf{x}_n + \sum_{m=1}^{N} p(m|\mathbf{x}_n)\mathbf{x}_m \right) \tag{5}$$

and its projection on the constraint space is given by zeroing its entries having indices in $\mathcal{P}$; call $\Pi_\mathcal{P}$ this projection operator. Then, we have the following step of length $\alpha \geq 0$ along the projected gradient:

$$\mathbf{X}^{(\tau+1)} = \mathbf{X}^{(\tau)} + \alpha \Pi_\mathcal{P}(\nabla_\mathbf{X} E(\mathbf{X}^{(\tau)})) \iff \mathbf{X}_\mathcal{M}^{(\tau+1)} = \mathbf{X}_\mathcal{M}^{(\tau)} + \alpha \left( \Pi_\mathcal{P}(\nabla_\mathbf{X} E(\mathbf{X}^{(\tau)})) \right)_\mathcal{M} \tag{6}$$

which updates only the missing entries $\mathbf{X}_\mathcal{M}$. Since our search direction is ascent and makes an angle with the gradient that is bounded away from $\pi/2$, and $E$ is lower bounded, continuously differentiable and has bounded Hessian (thus a Lipschitz continuous gradient) in $\mathbb{R}^{NL}$, by carrying out a line search that satisfies the Wolfe conditions, we are guaranteed convergence to a local stationary point, typically a maximizer [18, th. 3.2]. However, as reasoned later, we do not perform a line search at all, instead we fix the step size to the GBMS self-adapting step size, which results in a simple and faster algorithm consisting of carrying out a GBMS step on $\mathbf{X}$ (i.e., $\mathbf{X}^{(\tau+1)} = \mathbf{X}^{(\tau)} \mathbf{P}(\mathbf{X}^{(\tau)})$) and then refilling $\mathbf{X}_\mathcal{P}$ to the present values. While we describe the algorithm in this way for ease of explanation, in practice we do not actually compute the GBMS step for all $x_{dn}$ values, but only for the missing ones, which is all we need. Thus, our algorithm carries out GBMS denoising steps *within the missing-data subspace*. We can derive this result in a different way by starting from the unconstrained optimization problem $\max_{\mathbf{X}_\mathcal{P}} E(\mathbf{X}) = \sum_{n,m=1}^{N} G_\sigma(\mathbf{x}_n, \mathbf{x}_m)$ (equivalent to (4)), computing its gradient wrt $\mathbf{X}_\mathcal{P}$, equating it to zero and rearranging (in the same way the mean-shift algorithm is derived) to obtain a fixed-point iteration identical to our update above.

Fig. 1 shows the pseudocode for our denoising-based matrix completion algorithms (using three nonparametric denoising algorithms: GBMS, MBMS and LTP).

**Convergence and stopping criterion**   As noted above, we have guaranteed convergence by simply satisfying standard line search conditions, but a line search is costly. At present we do not have a proof that the GBMS step size satisfies such conditions, or indeed that the new iterate $\mathbf{X}_\mathcal{M}^{(\tau+1)}$ increases or leaves unchanged the objective, although we have never encountered a counterexample. In fact, it turns out that none of the work about GBMS that we know about proves that either: [10] proves that $\varnothing(\mathbf{X}^{(\tau+1)}) \leq \varnothing(\mathbf{X}^{(\tau)})$ for $0 < \rho < 1$, where $\varnothing(\cdot)$ is the set diameter, while [8, 12]

notes that $\mathbf{P}(\mathbf{X})$ has a single eigenvalue of value 1 and all others of magnitued less than 1. While this shows that all points converge to the same location, which indeed is the global maximum of (3), it does not necessarily follow that each step decreases $E$.

---

GBMS $(k, \sigma)$ with full or $k$-nn graph: given $\mathbf{X}_{D \times N}, \mathcal{M}$
**repeat**
  **for** $n = 1, \dots, N$
    $\mathcal{N}_n \leftarrow \{1, \dots, N\}$ (full graph) or
        $k$ nearest neighbors of $\mathbf{x}_n$ ($k$-nn graph)
    $\partial \mathbf{x}_n \leftarrow -\mathbf{x}_n + \sum_{m \in \mathcal{N}_n} \frac{G_\sigma(\mathbf{x}_n, \mathbf{x}_m)}{\sum_{m' \in \mathcal{N}_n} G_\sigma(\mathbf{x}_n, \mathbf{x}_{m'})} \mathbf{x}_m$     mean-shift step
  **end**
  $\mathbf{X}_{\mathcal{M}} \leftarrow \mathbf{X}_{\mathcal{M}} + (\partial \mathbf{X})_{\mathcal{M}}$     move points' missing entries
**until** validation error increases
**return** $\mathbf{X}$

---

MBMS $(L, k, \sigma)$ with full or $k$-nn graph: given $\mathbf{X}_{D \times N}, \mathcal{M}$
**repeat**
  **for** $n = 1, \dots, N$
    $\mathcal{N}_n \leftarrow \{1, \dots, N\}$ (full graph) or
        $k$ nearest neighbors of $\mathbf{x}_n$ ($k$-nn graph)
    $\partial \mathbf{x}_n \leftarrow -\mathbf{x}_n + \sum_{m \in \mathcal{N}_n} \frac{G_\sigma(\mathbf{x}_n, \mathbf{x}_m)}{\sum_{m' \in \mathcal{N}_n} G_\sigma(\mathbf{x}_n, \mathbf{x}_{m'})} \mathbf{x}_m$     mean-shift step
    $\mathcal{X}_n \leftarrow k$ nearest neighbors of $\mathbf{x}_n$
    $(\boldsymbol{\mu}_n, \mathbf{U}_n) \leftarrow \mathrm{PCA}(\mathcal{X}_n, L)$     estimate $L$-dim tangent space at $\mathbf{x}_n$
    $\partial \mathbf{x}_n \leftarrow (\mathbf{I} - \mathbf{U}_n \mathbf{U}_n^T) \partial \mathbf{x}_n$     subtract parallel motion
  **end**
  $\mathbf{X}_{\mathcal{M}} \leftarrow \mathbf{X}_{\mathcal{M}} + (\partial \mathbf{X})_{\mathcal{M}}$     move points' missing entries
**until** validation error increases
**return** $\mathbf{X}$

---

LTP $(L, k)$ with $k$-nn graph: given $\mathbf{X}_{D \times N}, \mathcal{M}$
**repeat**
  **for** $n = 1, \dots, N$
    $\mathcal{X}_n \leftarrow k$ nearest neighbors of $\mathbf{x}_n$
    $(\boldsymbol{\mu}_n, \mathbf{U}_n) \leftarrow \mathrm{PCA}(\mathcal{X}_n, L)$     estimate $L$-dim tangent space at $\mathbf{x}_n$
    $\partial \mathbf{x}_n \leftarrow (\mathbf{I} - \mathbf{U}_n \mathbf{U}_n^T)(\boldsymbol{\mu}_n - \mathbf{x}_n)$     project point onto tangent space
  **end**
  $\mathbf{X}_{\mathcal{M}} \leftarrow \mathbf{X}_{\mathcal{M}} + (\partial \mathbf{X})_{\mathcal{M}}$     move points' missing entries
**until** validation error increases
**return** $\mathbf{X}$

---

Figure 1: Our denoising matrix completion algorithms, based on Manifold Blurring Mean Shift (MBMS) and its particular cases Local Tangent Projection (LTP, $k$-nn graph, $\sigma = \infty$) and Gaussian Blurring Mean Shift (GBMS, $L = 0$); see [5] for details. $\mathcal{N}_n$ contains all $N$ points (full graph) or only $\mathbf{x}_n$'s nearest neighbors ($k$-nn graph). The index $\mathcal{M}$ selects the components of its input corresponding to missing values. Parameters: denoising scale $\sigma$, number of neighbors $k$, local dimensionality $L$.

However, the question of convergence as $\tau \to \infty$ has no practical interest in a denoising setting, because achieving a total denoising almost never yields a good matrix completion. What we want is to achieve *just enough* denoising and stop the algorithm, as was the case with GBMS clustering, and as is the case in algorithms for image denoising. We propose to determine the optimal number of iterations, as well as the bandwidth $\sigma$ and any other parameters, by cross-validation. Specifically, we select a held-out set by picking a random subset of the present entries and considering them as missing; this allows us to evaluate an error between our completion for them and the ground truth. We stop iterating when this error increases.

This argument justifies an algorithmic, as opposed to an optimization, view of denoising-based matrix completion: *apply a denoising step, refill the present values, iterate until the validation error increases*. This allows very general definitions of denoising, and indeed a low-rank projection is a form of denoising where points are not allowed outside the linear manifold. Our formulation using the objective function (4) is still useful in that it connects our denoising assumption with the more usual low-rank assumption that has been used in much matrix completion work, and justifies the refilling step as resulting from the present-data constraints under a gradient-projection optimization.

**MBMS denoising for matrix completion**   Following our algorithmic-based approach to denoising, we could consider generalized GBMS steps of the form $\tilde{\mathbf{X}} = \mathbf{X} \phi(\mathbf{P}(\mathbf{X}))$. For clustering, Carreira-Perpiñán [12] found an overrelaxed explicit step $\phi(\mathbf{P}) = (1 - \eta)\mathbf{I} + \eta\mathbf{P}$ with $\eta \approx 1.25$ to achieve similar clusterings but faster. Here, we focus instead on the MBMS variant of GBMS that allows only for orthogonal, not tangential, point motions (defined wrt their local tangent space as estimated by local PCA), with the goal of preserving low-dimensional manifold structure. MBMS has 3 user parameters: the bandwidth $\sigma$ (for denoising), and the latent dimensionality $L$ and the

number of neighbors $k$ (for the local tangent space and the neighborhood graph). A special case of MBMS called *local tangent projection (LTP)* results by using a neighborhood graph and setting $\sigma = \infty$ (so only two user parameters are needed: $L$ and $k$). LTP can be seen as doing a low-rank matrix completion locally. LTP was found in [5] to have nearly as good performance as the best $\sigma$ in several problems. MBMS also includes as particular cases GBMS ($L = 0$), PCA ($k = N$, $\sigma = \infty$), and no denoising ($\sigma = 0$ or $L = D$).

Note that if we apply MBMS to a dataset that lies on a linear manifold of dimensionality $d$ using $L \geq d$ then no denoising occurs whatsoever because the GBMS updates lie on the $d$-dimensional manifold and are removed by the corrector step. In practice, even if the data are assumed noiseless, the reconstruction from a low-rank method will lie close to but not exactly on the $d$-dimensional manifold. However, this suggests using largish ranks for the low-rank method used to reconstruct $\mathbf{X}$ and lower $L$ values in the subsequent MBMS run.

In summary, this yields a matrix completion algorithm where we apply an MBMS step, refill the present values, and iterate until the validation error increases. Again, in an actual implementation we compute the MBMS step only for the missing entries of $\mathbf{X}$. The shrinking problem of GBMS is less pronounced in our matrix completion setting, because we constrain some values not to change. Still, in agreement with [5], we find MBMS to be generally superior to GBMS.

**Computational cost**   With a full graph, the cost per iteration of GBMS and MBMS is $\mathcal{O}(N^2 D)$ and $\mathcal{O}(N^2 D + N(D + k)\min(D, k)^2)$, respectively. In practice with high-dimensional data, best denoising results are obtained using a neighborhood graph [5], so that the sums over points in eqs. (3) or (4) extend only to the neighbors. With a $k$-nearest-neighbor graph and if we do not update the neighbors at each iteration (which affects the result little), the respective cost per iteration is $\mathcal{O}(NkD)$ and $\mathcal{O}(NkD + N(D + k)\min(D, k)^2)$, thus linear in $N$. The graph is constructed on the initial $\mathbf{X}$ we use, consisting of the present values and an imputation for the missing ones achieved with a standard matrix completion method, and has a one-off cost of $\mathcal{O}(N^2 D)$. The cost when we have a fraction $\mu = \frac{|\mathcal{M}|}{ND} \in [0, 1]$ of missing data is simply the above times $\mu$. Hence the run time of our mean-shift-based matrix completion algorithms is faster the more present data we have, and thus faster than the usual GBMS or MBMS case, where all data are effectively missing.

## 3   Experimental results

We compare with representative methods of several approaches: a low-rank matrix completion method, singular value projection (SVP [7], whose performance we found similar to that of alternating least squares, ALS [3, 4]); fitting a $D$-dimensional Gaussian model with EM and imputing the missing values of each $\mathbf{x}_n$ as the conditional mean $\mathrm{E}\left\{\mathbf{x}_{n,\mathcal{M}_n}|\mathbf{x}_{n,\mathcal{P}_n}\right\}$ (we use the implementation of [19]); and the nonlinear method of [20] (nlPCA). We initialize GBMS and MBMS from some or all of these algorithms. For methods with user parameters, we set them by cross-validation in the following way: we randomly select 10% of the present entries and pretend they are missing as well, we run the algorithm on the remaining 90% of the present values, and we evaluate the reconstruction at the 10% entries we kept earlier. We repeat this over different parameters' values and pick the one with lowest reconstruction error. We then run the algorithm with these parameters values on the entire present data and report the (test) error with the ground truth for the missing values.

**100D Swissroll**   We created a 3D swissroll data set with $3\,000$ points and lifted it to 100D with a random orthonormal mapping, and added a little noise (spherical Gaussian with stdev 0.1). We selected uniformly at random 6.76% of the entries to be present. We use the Gaussian model and SVP (fixed rank = 3) as initialization for our algorithm. We typically find that these initial $\mathbf{X}$ are very noisy (fig. 3), with some reconstructed points lying between different branches of the manifold and causing a big reconstruction error. We fixed $L = 2$ (the known dimensionality) for MBMS and cross-validated the other parameters: $\sigma$ and $k$ for MBMS and GBMS (both using $k$-nn graph), and the number of iterations $\tau$ to be used. Table 1 gives the performance of MBMS and GBMS for testing, along with their optimal parameters. Fig. 3 shows the results of different methods at a few iterations. MBMS initialized from the Gaussian model gives the most remarkable denoising effect. To show that there is a wide range of $\sigma$ and number of iterations $\tau$ that give good performance with GBMS and MBMS, we fix $k = 50$ and run the algorithm with varying $\sigma$ values and plot the reconstruction error for missing entries over iterations in fig. 2. Both GBMS can achieve good

| Methods | RSSE | mean | stdev |
|---|---|---|---|
| Gaussian | 168.1 | 2.63 | 1.59 |
| + GBMS ($\infty$, 10, 0, 1) | 165.8 | 2.57 | 1.61 |
| + MBMS (1, 20, 2, 25) | 157.2 | 2.36 | 1.63 |
| SVP | 156.8 | 1.94 | 2.10 |
| + GBMS (3, 50, 0, 1) | 151.4 | 1.89 | 2.02 |
| + MBMS (3, 50, 2, 2) | 151.8 | 1.87 | 2.05 |

Table 1: Swissroll data set: reconstruction errors obtained by different algorithms along with their optimal parameters ($\sigma$, $k$, $L$, no. iterations $\tau$). The three columns show the root sum of squared errors on missing entries, the mean, and the standard deviation of the pointwise reconstruction error, resp.

| Methods | RSSE | mean | stdev |
|---|---|---|---|
| nlPCA | 7.77 | 26.1 | 42.6 |
| SVP | 6.99 | 21.8 | 39.3 |
| + GBMS (400,140,0,1) | 6.54 | 18.8 | 37.7 |
| + MBMS (500,140,9,5) | 6.03 | 17.0 | 34.9 |

Table 2: MNIST-7 data set: errors of the different algorithms and their optimal parameters ($\sigma$, $k$, $L$, no. iterations $\tau$). The three columns show the root sum of squared errors on missing entries ($\times 10^{-4}$), the mean, and the standard deviation of pixel errors, respectively.

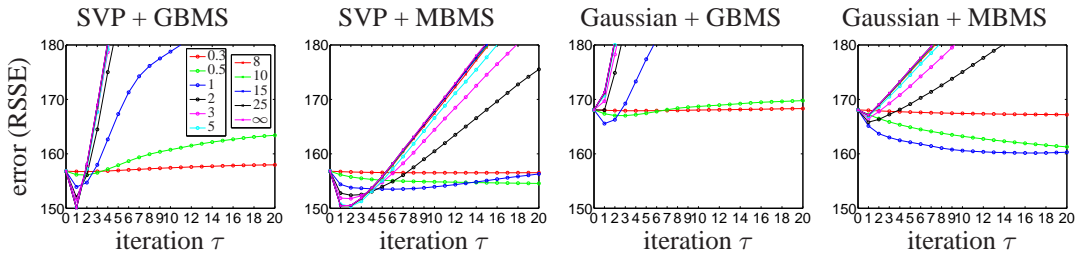

Figure 2: Reconstruction error of GBMS/MBMS over iterations (each curve is a different $\sigma$ value).

denoising (and reconstruction), but MBMS is more robust, with good results occurring for a wide range of iterations, indicating it is able to preserve the manifold structure better.

**Mocap data** We use the running-motion sequence 09_01 from the CMU mocap database with 148 samples ($\approx 1.7$ cycles) with 150 sensor readings (3D positions of 50 joints on a human body). The motion is intrinsically 1D, tracing a loop in 150D. We compare nlPCA, SVP, the Gaussian model, and MBMS initialized from the first three algorithms. For nlPCA, we do a grid search for the weight decay coefficient while fixing its structure to be $2 \times 10 \times 150$ units, and use an early stopping criterion. For SVP, we do grid search on $\{1, 2, 3, 5, 7, 10\}$ for the rank. For MBMS ($L = 1$) and GBMS ($L = 0$), we do grid search for $\sigma$ and $k$.

We report the reconstruction error as a function of the proportion of missing entries from 50% to 95%. For each missing-data proportion, we randomly select 5 different sets of present values and run all algorithms for them. Fig. 4 gives the mean errors of all algorithms. All methods perform well when missing-data proportion is small. nlPCA, being prone to local optima, is less stable than SVP and the Gaussian model, especially when the missing-data proportion is large. The Gaussian model gives the best and most stable initialization. At 95%, all methods fail to give an acceptable reconstruction, but up to 90% missing entries, MBMS and GBMS always beat the other algorithms. Fig. 4 shows selected reconstructions from all algorithms.

**MNIST digit '7'** The MNIST digit '7' data set contains 6 265 greyscale (0–255) images of size $28 \times 28$. We create missing entries in a way reminiscent of run-length errors in transmission. We generate 16 to 26 rectangular boxes of an area approximately 25 pixels at random locations in each image and use them to black out pixels. In this way, we create a high dimensional data set (784 dimensions) with about 50% entries missing on average. Because of the loss of spatial correlations within the blocks, this missing data pattern is harder than random.

The Gaussian model cannot handle such a big data set because it involves inverting large covariance matrices. nlPCA is also very slow and we cannot afford cross-validating its structure or the weight decay coefficient, so we picked a reasonable structure ($10 \times 30 \times 784$ units), used the default weight decay parameter in the code ($10^{-3}$), and allowed up to 500 iterations. We only use SVP as initialization for our algorithm. Since the intrinsic dimension of MNIST is suspected to be not very high,

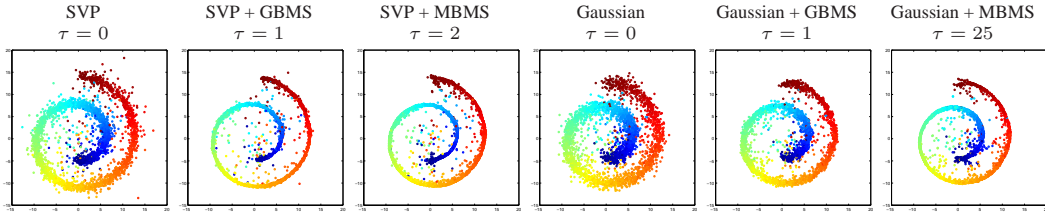

| SVP | SVP + GBMS | SVP + MBMS | Gaussian | Gaussian + GBMS | Gaussian + MBMS |
| $\tau = 0$ | $\tau = 1$ | $\tau = 2$ | $\tau = 0$ | $\tau = 1$ | $\tau = 25$ |

Figure 3: Denoising effect of the different algorithms. For visualization, we project the 100D data to 3D with the projection matrix used for creating the data. Present values are refilled for all plots.

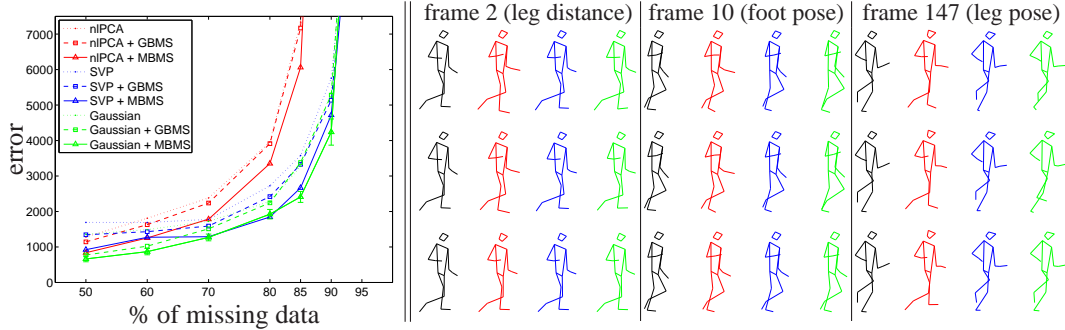

Figure 4: *Left*: mean of errors (RSSE) of 5 runs obtained by different algorithms for varying percentage of missing values. Errorbars shown only for Gaussian + MBMS to avoid clutter. *Right*: sample reconstructions when 85% percent data is missing. *Row 1*: initialization. *Row 2*: init+GBMS. *Row 3*: init+MBMS. Color indicates different initialization: black, original data; red, nlPCA; blue, SVP; green, Gaussian.

we used rank 10 for SVP and $L = 9$ for MBMS. We also use the same $k = 140$ as in [5]. So we only had to choose $\sigma$ and the number of iterations via cross-validation.

Table 2 shows the methods and their corresponding error. Fig. 5 shows some representative reconstructions from different algorithms, with present values refilled. The mean-shift averaging among closeby neighbors (a soft form of majority voting) helps to eliminate noise, unusual strokes and other artifacts created by SVP, which by their nature tend to occur in different image locations over the neighborhood of images.

## 4   Related work

Matrix completion is widely studied in theoretical compressed sensing [1, 2] as well as practical recommender systems [3, 4]. Most matrix completion models rely on a low-rank assumption, and cannot fully exploit a more complex structure of the problem, such as curved manifolds. Related work is on multi-task learning in a broad sense, which extracts the common structure shared by multiple related objects and achieves simultaneous learning on them. This includes applications such as alignment of noise-corrupted images [21], recovery of images with occlusion [22], and even learning of multiple related regressors or classifiers [23]. Again, all these works are essentially based on a subspace assumption, and do not generalize to more complex situations.

A line of work based on a nonlinear low-rank assumption (with a latent variable $\mathbf{z}$ of dimensionality $L < D$) involves setting up a least-squares error function $\min_{\mathbf{f}, \mathbf{z}} \sum_{n=1}^{N} \|\mathbf{x}_n - \mathbf{f}(\mathbf{z}_n)\|^2 = \sum_{n,d=1}^{N,D} (x_{dn} - f_d(\mathbf{z}_n))^2$ where one ignores the terms for which $x_{dn}$ is missing, and estimates the function $\mathbf{f}$ and the low-dimensional data projections $\mathbf{Z}$ by alternating optimization. Linear functions $\mathbf{f}$ have been used in the homogeneity analysis literature [24], where this approach is called "missing data deleted". Nonlinear functions $\mathbf{f}$ have been used recently (neural nets [20]; Gaussian processes for collaborative filtering [25]). Better results are obtained if adding a projection term $\sum_{n=1}^{N} \|\mathbf{z}_n - \mathbf{F}(\mathbf{x}_n)\|^2$ and optimizing over the missing data as well [26].

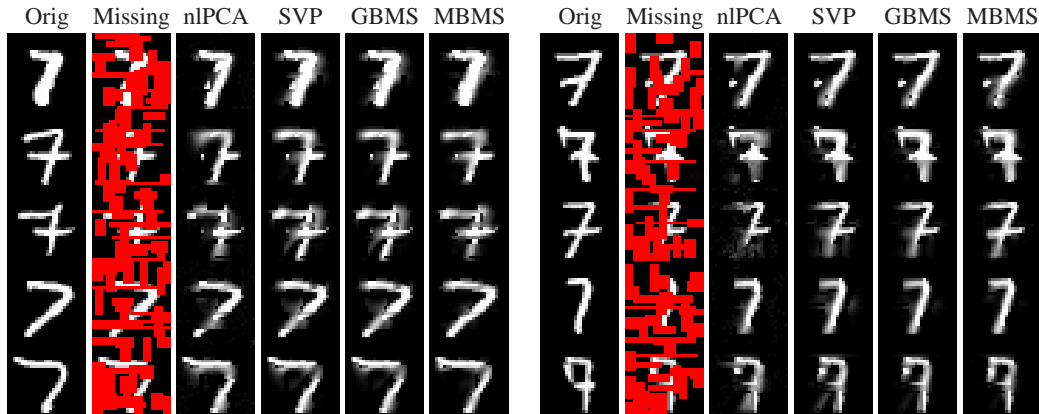

Figure 5: Selected reconstructions of MNIST block-occluded digits '7' with different methods.

Prior to our denoising-based work there have been efforts to extend the low-rank models to smooth manifolds, mostly in the context of compressed sensing. Baraniuk and Wakin [27] show that certain random measurements, e.g. random projection to a low-dimensional subspace, can preserve the metric of the manifold fairly well, if the intrinsic dimension and the curvature of the manifold are both small enough. However, these observations are not suitable for matrix completion and no algorithm is given for recovering the signal. Chen et al. [28] explicitly model a pre-determined manifold, and use this to regularize the signal when recovering the missing values. They estimate the manifold given complete data, while no complete data is assumed in our matrix completion setting. Another related work is [29], where the manifold modeled with Isomap is used in estimating the positions of satellite cameras in an iterative manner.

Finally, our expectation that the value of a missing entry can be predicted from the values of neighboring points is similar to one category of collaborative filtering methods that essentially use similar users/items to predict missing values [3, 4].

## 5 Conclusion

We have proposed a new paradigm for matrix completion, denoising, which generalizes the commonly used assumption of low rank. Assuming low-rank implies a restrictive form of denoising where the data is forced to have zero variance away from a linear manifold. More general definitions of denoising can potentially handle data that lives in a low-dimensional manifold that is nonlinear, or whose dimensionality varies (e.g. a set of manifolds), or that does not have low rank at all, and naturally they handle noise in the data. Denoising works because of the fundamental fact that a missing value can be predicted by averaging nearby present values.

Although we motivate our framework from a constrained optimization point of view (*denoise subject to respecting the present data*), we argue for an algorithmic view of denoising-based matrix completion: *apply a denoising step, refill the present values, iterate until the validation error increases*. In turn, this allows different forms of denoising, such as based on low-rank projection (earlier work) or local averaging with blurring mean-shift (this paper). Our nonparametric choice of mean-shift averaging further relaxes assumptions about the data and results in a simple algorithm with very few user parameters that afford user control (denoising scale, local dimensionality) but can be set automatically by cross-validation. Our algorithms are intended to be used as a postprocessing step over a user-provided initialization of the missing values, and we show they consistently improve upon existing algorithms.

The MBMS-based algorithm bridges the gap between pure denoising (GBMS) and local low rank. Other definitions of denoising should be possible, for example using temporal as well as spatial neighborhoods, and even applicable to discrete data if we consider denoising as a majority voting among the neighbours of a vector (with suitable definitions of votes and neighborhood).

**Acknowledgments** Work supported by NSF CAREER award IIS–0754089.

# References

[1] Emmanuel J. Candès and Benjamin Recht. Exact matrix completion via convex optimization. *Foundations of Computational Mathematics*, 9(6):717–772, December 2009.

[2] Emmanuel J. Candès and Terence Tao. The power of convex relaxation: Near-optimal matrix completion. *IEEE Trans. Information Theory*, 56(5):2053–2080, April 2010.

[3] Yehuda Koren. Factorization meets the neighborhood: A multifaceted collaborative filtering model. *SIGKDD 2008*, pages 426–434, Las Vegas, NV, August 24–27 2008.

[4] Robert Bell and Yehuda Koren. Scalable collaborative filtering with jointly derived neighborhood interpolation weights. *ICDM 2007*, pages 43–52, October 28–31 2007.

[5] Weiran Wang and Miguel Á. Carreira-Perpiñán. Manifold blurring mean shift algorithms for manifold denoising. *CVPR 2010*, pages 1759–1766, San Francisco, CA, June 13–18 2010.

[6] Matthias Hein and Markus Maier. Manifold denoising. *NIPS 2006*, 19:561–568. MIT Press, 2007.

[7] Prateek Jain, Raghu Meka, and Inderjit S. Dhillon. Guaranteed rank minimization via singular value projection. *NIPS 2010*, 23:937–945. MIT Press, 2011.

[8] Miguel Á. Carreira-Perpiñán. Fast nonparametric clustering with Gaussian blurring mean-shift. *ICML 2006*, pages 153–160. Pittsburgh, PA, June 25–29 2006.

[9] Keinosuke Fukunaga and Larry D. Hostetler. The estimation of the gradient of a density function, with application in pattern recognition. *IEEE Trans. Information Theory*, 21(1):32–40, January 1975.

[10] Yizong Cheng. Mean shift, mode seeking, and clustering. *IEEE Trans. PAMI*, 17(8):790–799, 1995.

[11] Dorin Comaniciu and Peter Meer. Mean shift: A robust approach toward feature space analysis. *IEEE Trans. PAMI*, 24(5):603–619, May 2002.

[12] Miguel Á. Carreira-Perpiñán. Generalised blurring mean-shift algorithms for nonparametric clustering. *CVPR 2008*, Anchorage, AK, June 23–28 2008.

[13] Gabriel Taubin. A signal processing approach to fair surface design. *SIGGRAPH 1995*, pages 351–358.

[14] Mathieu Desbrun, Mark Meyer, Peter Schröder, and Alan H. Barr. Implicit fairing of irregular meshes using diffusion and curvature flow. *SIGGRAPH 1999*, pages 317–324.

[15] Fan R. K. Chung. *Spectral Graph Theory*. American Mathematical Society, Providence, RI, 1997.

[16] Jianbo Shi and Jitendra Malik. Normalized cuts and image segmentation. *IEEE Trans. PAMI*, 22(8):888–905, August 2000.

[17] Roger A. Horn and Charles R. Johnson. *Matrix Analysis*. Cambridge University Press, 1986.

[18] Jorge Nocedal and Stephen J. Wright. *Numerical Optimization*. Springer-Verlag, New York, second edition, 2006.

[19] Tapio Schneider. Analysis of incomplete climate data: Estimation of mean values and covariance matrices and imputation of missing values. *Journal of Climate*, 14(5):853–871, March 2001.

[20] Matthias Scholz, Fatma Kaplan, Charles L. Guy, Joachim Kopka, and Joachim Selbig. Non-linear PCA: A missing data approach. *Bioinformatics*, 21(20):3887–3895, October 15 2005.

[21] Yigang Peng, Arvind Ganesh, John Wright, Wenli Xu, and Yi Ma. RASL: Robust alignment by sparse and low-rank decomposition for linearly correlated images. *CVPR 2010*, pages 763–770, 2010.

[22] A. M. Buchanan and A. W. Fitzgibbon. Damped Newton algorithms for matrix factorization with missing data. *CVPR 2005*, pages 316–322, San Diego, CA, June 20–25 2005.

[23] Andreas Argyriou, Theodoros Evgeniou, and Massimiliano Pontil. Multi-task feature learning. *NIPS 2006*, 19:41–48. MIT Press, 2007.

[24] Albert Gifi. *Nonlinear Multivariate Analysis*. John Wiley & Sons, 1990.

[25] Neil D. Lawrence and Raquel Urtasun. Non-linear matrix factorization with Gaussian processes. *ICML 2009*, Montreal, Canada, June 14–18 2009.

[26] Miguel Á. Carreira-Perpiñán and Zhengdong Lu. Manifold learning and missing data recovery through unsupervised regression. *ICDM 2011*, December 11–14 2011.

[27] Richard G. Baraniuk and Michael B. Wakin. Random projections of smooth manifolds. *Foundations of Computational Mathematics*, 9(1):51–77, February 2009.

[28] Minhua Chen, Jorge Silva, John Paisley, Chunping Wang, David Dunson, and Lawrence Carin. Compressive sensing on manifolds using a nonparametric mixture of factor analyzers: Algorithm and performance bounds. *IEEE Trans. Signal Processing*, 58(12):6140–6155, December 2010.

[29] Michael B. Wakin. A manifold lifting algorithm for multi-view compressive imaging. In *Proc. 27th Conference on Picture Coding Symposium (PCS'09)*, pages 381–384, 2009.

